# Designing Linear Threshold Based Neural Network Pattern Classifiers

Terrence L. Fine
School of Electrical Engineering
Cornell University
Ithaca, NY 14853

## Abstract

The three problems that concern us are identifying a natural domain of pattern classification applications of feedforward neural networks, selecting an appropriate feedforward network architecture, and assessing the tradeoff between network complexity, training set size, and statistical reliability as measured by the probability of incorrect classification. We close with some suggestions, for improving the bounds that come from Vapnik-Chervonenkis theory, that can narrow, but not close, the chasm between theory and practice.

## 1 Speculations on Neural Network Pattern Classifiers

(1)     The goal is to provide rapid, reliable classification of new inputs from a pattern source. Neural networks are appropriate as pattern classifiers when the pattern sources are ones of which we have little understanding, beyond perhaps a nonparametric statistical model, but we have been provided with classified samples of features drawn from each of the pattern categories. Neural networks should be able to provide rapid and reliable computation of complex decision functions. The issue in doubt is their statistical response to new inputs.

(2)     The pursuit of optimality is misguided in the context of Point (1). Indeed, it is unclear what might be meant by 'optimality' in the absence of a more detailed mathematical framework for the pattern source.

(3)     The well-known, oft-cited 'curse of dimensionality' exposed by Richard Bellman may be a 'blessing' to neural networks. Individual network processing nodes (e.g., linear threshold units) become more powerful as the number of their inputs increases. For a large enough number $n$ of points in an input space of $d$ dimensions, the number of dichotomies that can be generated by such a node grows exponentially in $d$. This suggests that, unlike all previous efforts at pattern classification that required substantial effort directed at the selection of low-dimensional feature vectors so as to make the decision rule calculable, we may now be approaching a

position from which we can exploit raw data (e.g., the actual samples in a time series or pixel values in an image). Even if we are as yet unable to achieve this, it is clear from the reports on actual pattern classifiers that have been presented at NIPS90 and the accompanying Keystone Workshop that successful neural network pattern classifiers have been constructed that accept as inputs feature vectors having hundreds of components (e.g., Guyon, et al. [1990]).

(4)    The blessing of dimensionality is not granted if there is either a large subset of critically important components that will force the network to be too complex or a small subset that contains almost all of the information needed for accurate discrimination. The network is liable to be successful in those cases where the input or feature vector $x$ has components that are individually nearly irrelevant, although collectively they enable us to discriminate well. Examples of such feature vectors might be the responses of individual fibers in the optic nerve, a pixel array for an image of an alphanumeric character, or the set of time samples of an acoustic transient. No one fiber, pixel value, or time sample provides significant information as to the true pattern category, although all of them taken together may enable us to do nearly error-free classification. An example in which all components are critically important is the calculation of parity. On our account, this is the sort of problem for which neural networks are inappropriate, albeit it has been repeatedly established that they can calculate parity.

We interpret 'critically important' very weakly as meaning that the subspace spanned by the subset of critically important features/inputs needs to be partitioned by the classifier so that there is at least one bounded region. If the nodes are linear threshold units then to carve out a bounded region, minimally a simplex, in a subspace of dimension $c$, where $c$ is the size of the subset of critically important inputs, will require a network having at least $c + 1$ nodes in the first layer.

(5)    Neural networks have opened up a new application domain wherein in practice we can intelligently construct nonlinear pattern classifiers characterized by thousands of parameters. In practice, nonlinear statistical models, ones not defined in terms of a covariance matrix, seem to be restricted to a few parameters.

(6)    Nonetheless, Occam's Razor advises us to be sparing of parameters. We should be particularly cautious about the problem of overfitting when the number of parameters in the network is not much less than the number of training samples. Theory needs to provide practice with better insight and guidelines for avoiding overfitting and for the use of restrictions on training time as a guard against overfitting a system with almost as many adjustable parameters as there are data points.

(7)    Points (1) and (5) combine to suggest that analytical approaches to network performance evaluation based upon typical statistical ideas may either be difficult to carry out or yield conclusions of little value to practice. There is no mismatch between statistical theory and neural networks in principle, but there does seem to be a significant mismatch in practice. While we are usually dealing with thousands of training samples, the complexity of the network means that we are not in a regime where asymptotic analyses (large sample behavior) will prove informative. On the other hand, the problem is far to complex to be resolved by 'exact' small sample analyses. These considerations serve to validate the widespread use of simulation studies to assess network design and performance.

## 2 The QED Architecture

### 2.1 QED Overview

One may view a classifier as either making the decision as to the correct class or as providing 'posterior' probabilities for the various classes. If we adopt the latter approach, then the use of sigmoidal units having a continuum of responses is appropriate. If, however, we adopt the first approach, then we require hard-limiting devices to select one of only finitely many (in our case only two) pattern classes. This is the approach that we adopt and it leads us to reliance upon linear threshold units (LTUs).

We have focused our attention upon a flexible architecture consisting of a first hidden layer that is viewed as a *quantizer* of the input feature vector $\underline{x}$ and is therefore referred to as the Q-layer. The binary outputs from the Q-layer are then input to a second hidden layer whose function is to *expand* the dimension of the set of Q-layer outputs. The E-layer enables us to exploit the blessing of dimensionality in that by choosing it wide enough we can ensure that all Boolean functions of the binary outputs of the Q-layer are now implementable as linearly separable functions of the E-layer outputs. Hence, to implement a binary classifier we need a third layer consisting of only a single node to effect the desired *decision*, and this output layer is referred to as the D-layer. The layers taken together are called a QED architecture.

### 2.2 Constructing the Q-Layer

The first layer in a feedforward neural network having LTUs can always be viewed as a quantizer. Subsequent layers in the network only see the input $\underline{x}$ through the window provided by the first layer quantization. We do not expect to be able to quantize/partition the input space, say $\mathbf{R}^d$ for large $d$, into many small compact regions; to do so would require that $m \gg d$, as noted in Point (4) of the preceding section. Hence, asymptotic results drawn from deterministic approximation theory are unlikely to be helpful here. One might have recourse to the large literature on vector quantization (e.g., the special issue on quantization of the *IEEE Transactions on Information Theory*, March 1982), but we expect to quantize vectors of high dimension into a relatively small number of regions. Most of the information-theoretic literature on vector quantization does not address this domain of very low information rate (bits/coordinate). A more promising direction is that of clustering algorithms (e.g., k-means as in Pollard [1982], Darken and Moody [1990]) to guide the choice of Q-layer.

### 2.3 Constructing the E,D-Layers

Space limitations prevent us from detailed discussion of the formation of the E,D layers. In brief, the E-layer can be composed of $2^m$, often fewer, nodes where the weights to the $i$th node from the $m$ Q-layer nodes are a binary representation of the index $i$ with '0' replaced by '-1'. No training is required for the E-layer. The desired D-layer responses of 0 or 1 are formed simply by assigning weight $t$ to connections from E-layer nodes corresponding to input patterns from class $t$, and summing and thresholding at 1/2. The training set $\mathcal{T}$ must be consulted to determine, say, on

the basis of majority rule, the category $t \in \{0, 1\}$ to assign to a given E-layer node.

## 2.4   The Width of the Q-Layer

The overall complexity of the QED net depends upon the number $m$ of nodes in the Q-layer. Hence, our proposal will only be of practical interest if $m$ need not be large. As a first argument concerning the size of this parameter, if $m \leq d$ then $m$ hyperplanes in general position partition $\mathbf{R}^d$ into $2^m$ regions/cells. These cells are only of interest to us if we know how to assign them to pattern classes. From the perspective of Point (1) in the preceding section, we can only determine a classification of a cell if we have classified data points lying in the cell. Thus, if we wish to make rational use of $m$ nodes in the Q-layer, then we should have in excess of $2^m$ data points in our training set. If we have fewer data points in $\mathcal{T}$ then we will be generating a multitude of cells about whose categorization we know no more than that provided by possibly known prior class probabilities. Another estimate of the required sample size is obtained by assuming that data points are placed at random in the cells. In this case results summarized and improved on in Flatto [1982] suggest that we will need in excess of $m2^m$ points to have a reasonable probability of having all cells occupied by data points. Many of the experimental studies reported at the meeting and workshops of NIPS90 assumed training set sizes no larger than about 10,000, implying that we need not consider $m$ in excess of about 10. This number of nodes still yields a tractable QED architecture.

A second argument on which to base an *a priori* determination of $m$ can be made by considering the problem-average performance analyses carried out by Hughes [1968]. He found that the probability of correct classification for a randomly selected classification problem, with equal prior probabilities for selecting a class, varied with the number $M$ of possible feature values as $\frac{3M-2}{4M-2}$. This conclusion would suggest that a Q-layer containing as few as five properly selected nodes would suffice (Point (2)) for the design of a good pattern classifier.

In any event, both of our arguments suggest that a QED net having no more than about 10 Q-layer nodes might be adequate for many applications. At worst we would have to contemplate about 1,000 nodes in the E-layer, and this is not a prohibitively large number given current directions in hardware development. Nonetheless, the contradiction between our suggestions and current practice suggests that our conclusions are only tentative, and they need to be explored through applications, simulations, and studies of statistical generalization ability.

# 3   Sketch of Vapnik-Chervonenkis Theory of Statistical Generalization

We assume that there are two pattern classes labelled by $t \in \{0, 1\}$. A pattern sample is reduced by a preprocessor to a feature vector $\underline{x} \in \mathbf{R}^d$. Point (3) expresses the goal of having this reduction be significantly less than would be required by an approach that does not use neural networks. Neural networks are generically labelled by $\eta : \mathbf{R}^d \to \{0, 1\}$, $\eta(\underline{x}) = t$. $\mathcal{N} = \{\eta\}$ denotes the family of networks described by an architecture. As above, $m$ denotes the width of the first hidden

layer, and $M$ denotes the number of cells/regions into which a net in $\mathcal{N}$ can partition $\mathbf{R}^d$. Typically, $M = 2^m$. The training set $\mathcal{T} = \{(\underline{x}_i, t_i), i = 1, n\}$. We hypothesize that the elements of $\mathcal{T}$ are *i.i.d.* as $\mathcal{P}(\underline{x}, t)$, which is unknown.

Performance is measured by error probabilities,

$$\mathcal{E}(\eta) = \mathcal{P}(\eta(\underline{x}) \neq t).$$

A good (it need not be unique) net in the family $\mathcal{N}$ is

$$\eta^0 = \mathrm{argmin}_{\eta \in \mathcal{N}} \mathcal{E}(\eta), \qquad \mathcal{E}(\eta^0) = \min_{\eta \in \mathcal{N}} \mathcal{E}(\eta).$$

$\mathcal{E}_B$ denotes the Bayes error probability calculated on the basis of $\mathcal{P}(\underline{x}, t)$.

The empirical error frequency $\nu_{\mathcal{T}}(\eta)$ sustained by net $\eta$ applied to $\mathcal{T}$ is

$$\nu_{\mathcal{T}}(\eta) = \frac{1}{n} \sum_{i=1}^{n} |\eta(\underline{x}_i) - t_i|.$$

A net in $\mathcal{N}$ having good classification performance on the training set $\mathcal{T}$ is

$$\eta^* = \mathrm{argmin}_{\eta \in \mathcal{N}} \nu_{\mathcal{T}}(\eta).$$

By definition,

$$\mathcal{E}(\eta^*) \geq \mathcal{E}(\eta^0) \geq \mathcal{E}_B.$$

Let $m_{\mathcal{N}}(n)$ denote the VC growth function– the maximum, taken over all sets of $n$ points in the input space, of the number of subsets that can be generated by the classification functions in $\mathcal{N}$. Let $V_{\mathcal{N}}$ denote the VC capacity, the largest $n$ for which $\mathcal{N}$ can generate all $2^n$ of the subsets of some such set of $n$ points.

For $n > V_{\mathcal{N}}$, Vapnik-Chervonenkis theory (Vapnik [1982], Pollard [1984], Baum and Haussler [1989]) can be adapted to yield the VC upper bound

$$\mathcal{P}(\mathcal{E}(\eta^*) - \mathcal{E}(\eta^0) \geq \epsilon) \leq 6 \frac{(2n)^{V_{\mathcal{N}}}}{V_{\mathcal{N}}!} e^{-n\epsilon^2/16} = 6 e^{V_{\mathcal{N}} \log 2n - \log V_{\mathcal{N}}! - n\epsilon^2/16}.$$

Let $n_c$ denote the critical value of sample size $n$ for which the exponent first becomes negative. If $n < n_c$ then the upper bound will exceed unity and be uninformative. However, if $n > n_c$ then the upper bound will converge to zero exponentially fast in sample size. An approximate solution for $n_c$ from the VC upper bound yields

$$n_c \approx \frac{16}{\epsilon^2} V_{\mathcal{N}} \left( \log \frac{32e}{\epsilon^2} + \log \log \frac{32e}{\epsilon^2} \right).$$

If for purposes of illustration we take $\epsilon = .1, V_{\mathcal{N}} = 50$, then we find that $n_c \approx 902,000$. This conclusion, obtained by a direct application of Vapnik-Chervonenkis theory, disagrees by orders of magnitude with the experience of practitioners gained in training such low-complexity networks (about 50 connections).

## 4    Tightening the VC Argument

There are several components of the derivation of VC bounds that involve approximations and these, therefore, can be sources for improving these bounds. These

approximations include recourses to Chernoff/Hoeffding bounds, union bounds, estimates of $m_{\mathcal{N}}(n)$, and the relation between $\mathcal{E}(\eta^*) - \mathcal{E}(\eta^0)$ and $2\sup_\eta |\nu_T(\eta) - \mathcal{E}(\eta)|$. There is a belief among members of the neural network community that the weakness of the VC argument lies in the fact that by dealing with all possible underlying distributions $\mathcal{P}$ it is dealing with the worst case, and this worst case forces the large sample sizes. We agree with all but the last part of this belief. VC arguments being independent of the choice of $\mathcal{P}$ do indeed have to deal with worst cases. However, the worst case is dealt with through recourse to Chernoff/Hoeffding inequalities, and it is easily shown that these inequalities are not the source of our difficulties. A more promising direction in which to seek realistic estimates of training set size is through reductions in $m_{\mathcal{N}}(n)$ achieved through constraints on the architecture $\mathcal{N}$. One such restriction is through training time bounds that in effect restrict the portion of $\mathcal{N}$ that can be explored. Two other restrictions are discussed below.

## 5   Restricting the Architecture

### 5.1   Parameter Quantization

We can control the growth function contribution by quantizing all network parameters to $k$ bits and thereby restricting $\mathcal{N}$. The VC dimension of a LTU with parameters quantized to $k \geq 1$ bits equals the VC dimension of the LTU with real-valued parameters. Hence, VC arguments show no improvement. However, there are now only $2^{km(d+1)}$ distinct first layers of $m$ nodes accepting vectors from $\mathrm{R}^d$. Hence, there are no more than $2^{2^m + km(d+1)}$ QED nets, and the restricted $\mathcal{N}$ has only finitely many members.

Direct application of the union bound and Chernoff inequality yield

$$\mathcal{P}(\mathcal{E}(\eta^*) - \mathcal{E}(\eta^0) \geq \epsilon) \leq 2^{2 + 2^m + km(d+1)} e^{-n\epsilon^2/2}.$$

When $\epsilon = .1, m = 5, d = 10$ this bound becomes less than unity for $n > n_c = 4710 + 7625k$. Thus, even 1-bit quantization suggests a training sample size in excess of 4700 for reliable generalization of even this simple network.

### 5.2   Clustering

The growth function $m_{\mathcal{N}}(n)$ 'overestimates' the number of cases we need to be concerned with in dealing with the random variable $Z(\eta) = |\nu_T(\eta) - \nu_{T'}(\eta)|$ encountered in VC theory derivations. We are only interested in whether $Z$ exceeds a prescribed precision level $\epsilon$, and not whether, say, $Z(\eta_1)$ differs from $Z(\eta_2)$ by as little as $\frac{1}{n}$ due to $\eta_2$ disagreeing with $\eta_1$ at only a single sample point.

To enforce consideration of networks as being different only if they yield classifications of $T$ disagreeing substantially with each other we might proceed by clustering the points in $T$ into $\kappa$ clusters for each of the two classes. We then train the network so that decision boundaries do not subdivide individual clusters (see also Devroye and Wagner [1979]). The union bound and Chernoff inequality yield

$$\mathcal{P}(\mathcal{E}(\eta^*) - \mathcal{E}(\eta^0) \geq \epsilon) \leq 2^{2 + 4\kappa} e^{-n\epsilon^2/2},$$

a result that is independent of the input dimension $d$.

If we again choose $\epsilon = .1$ then the sample size $n$ required to make this upper bound less than unity is about $280 + 560\kappa$. For accuracy at the precision level $\epsilon$ we should expect to have $\kappa \geq 1/\epsilon$. Hence, the least acceptable sample size should exceed 5,880. If we hope to make full use of the capabilities of the net, then we should expect to have clusters in almost all of the $2^m$ cells. If we take this to mean that $2\kappa = 2^m$, then $n > 9,240$ for $m = 5$. If clusters were equally likely to fall into each of the $M$ cells then we would require $M(\log M + \alpha)$ clusters for a probability of no empty cell being approximately $e^{-e^{-\alpha}}$ (e.g., Flatto [1982]). Roughly, for $m = 5$ we should then aim for $2\kappa = 110$ and a sample size exceeding 31,000. Large as this estimate is, it is still a factor of 30 below what a direct application of VC theory yields.

## Acknowledgements

I wish to thank Thomas W. Parks for insightful remarks on several of the topics discussed above.

This paper was prepared with partial support from DARPA through AFOSR-90-0016A.

## References

Baum, E., D. Haussler [1989], What size net gives valid generalization?, in D. Touretzky, ed., *Advances in Neural Information Processing Systems 1*, Morgan Kaufman Pub., 81-90.

Darken, C., J. Moody [1990], Fast adaptive k-means clustering, NIPS90.

Devroye, L., T. Wagner [1979], Distribution-free bounds with the resubstitution error estimate, *IEEE Trans. on Information Theory*, **IT-25**, 208-210.

Flatto, L. [1982], Limit theorems for some random variables associated with urn models, *Annals of Probability*, **10**, 927-934.

Guyon, I., P. Albrecht, Y. Le Cun, J. Denker, W. Hubbard [1990], Design of a neural network character recognizer for a touch terminal, listed as to appear in *Pattern Recognition*, presented orally by Le Cun at the 1990 Keystone Workshop.

Hughes, G. [1968], On the mean accuracy of statistical pattern recognizers, *IEEE Trans. on Information Theory*, **14**, 55-63.

Pollard, D. [1982], A central limit theorem for k-means clustering, *Annals of Probability*, **10**, 919-926.

Pollard, D. [1984], *Convergence of Stochastic Processes*, Springer Verlag.

Vapnik, V. [1982], *Estimation of Dependences Based on Empirical Data*, Springer Verlag.